# A Model for Chemosensory Reception

**Rainer Malaka, Thomas Ragg**
Institut für Logik, Komplexität und Deduktionssysteme
Universität Karlsruhe, PO Box
D-76128 Karlsruhe, Germany
e-mail: malaka@ira.uka.de, ragg@ira.uka.de

**Martin Hammer**
Institut für Neurobiologie
Freie Universität Berlin
D-14195 Berlin, Germany
e-mail: mhammer@castor.zedat.fu-berlin.de

## Abstract

A new model for chemosensory reception is presented. It models reactions between odor molecules and receptor proteins and the activation of second messenger by receptor proteins. The mathematical formulation of the reaction kinetics is transformed into an artificial neural network (ANN). The resulting feed-forward network provides a powerful means for parameter fitting by applying learning algorithms. The weights of the network corresponding to chemical parameters can be trained by presenting experimental data. We demonstrate the simulation capabilities of the model with experimental data from honey bee chemosensory neurons. It can be shown that our model is sufficient to rebuild the observed data and that simpler models are not able to do this task.

## 1  INTRODUCTION

Terrestrial animals, vertebrates and invertebrates, have developed very similar solutions for the problem of recognizing volatile substances [Vogt *et al.*, 1989]. Odor molecules bind to receptor proteins (receptor sites) at the cell membrane of the sensory cell. This interaction of odor molecules and receptor proteins activates a G-protein mediated second

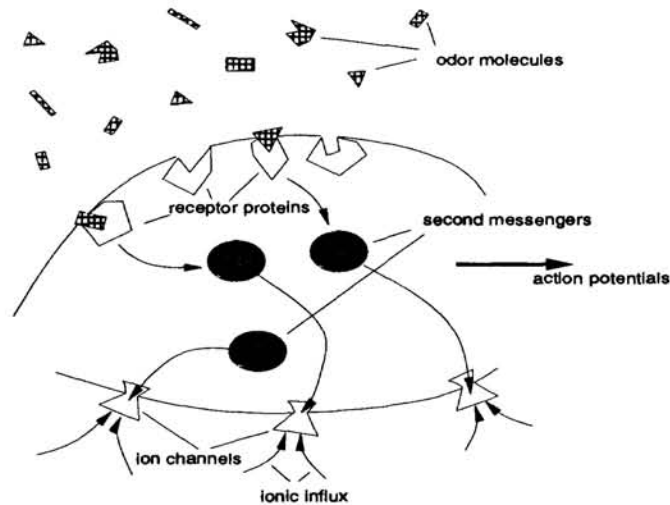

Figure 1: Reaction cascade in chemosensory neurons. Volatile odor molecules reach receptor proteins at the surface of the chemosensory neuron. The odor bound binding proteins activate second messengers (e.g. G-proteins). The activated second messengers cause a change in the conductivity of ion channels. Through ionic influx a depolarization can build an action potential.

messenger process. The concentrations of cAMP or $IP_3$ rise rapidly and activate cyclic-nucleotide-gated ion channels or $IP_3$-gated ion channels [Breer *et al.*, 1989, Shepherd, 1991]. As a result of this second messenger reaction cascade the conductivity of ion channels is changed and the cell can be hyperpolarized or depolarized, which can cause the generation of action potentials. It has been shown that one odor is able to activate different second messenger processes and that there is some interaction between the different second messenger processes [Breer & Boekhoff, 1992].

Figure 1 shows schematically the cascade of reactions from odor molecules over receptor proteins and second messengers up to the changing of ion channel conductance and the generation of action potentials.

Responses of sensory neurons can be very complex. The response as a function of the odor concentration is highly non-linear. The response to mixtures can be synergistic or inhibitory relative to the response to the components of the compound. A synergistic effect occurs, if the response of one sensory cell to a binary mixture of two odors $A_1$ and $A_2$ with concentrations $[A_1]$, $[A_2]$ is higher than the sum of the responses to the odors $A_1$, $A_2$ at concentrations $[A_1]$, $[A_2]$ alone. An inhibitory effect occurs, if the response to the mixture is smaller than either response to the single odors. In bee subplacode and placode recordings both effects can be observed [Akers & Getz, 1993].

Models of chemosensory reception should be complex enough to simulate the inhibitory and synergistic effects observed in sensory neurons, and they must provide a means for parameter fitting. We want to introduce a computational model which is constructed analogously to the chemical reaction cascade in the sensory neuron. The model can be expressed as an ANN and all unknown parameters can be trained with a learning algorithm.

## 2  THE RECEPTOR TRANSDUCER MODEL

The first step of odor reception is done by receptor proteins located at the cell membrane. There may be many receptor protein types in sensory cells at different concentrations and with different sensitivity to various odors. There is the possibility for different odors ligands $A_i$ to react with a receptor protein $R_j$, but it is also possible for a single odor to react with different receptor proteins.

The second step is the activation of second messengers. Ennis proposed a modelling of these complex reactions by a reaction step of activated odor-receptor complexes with transducer mechanisms [Ennis, 1991]. These transducers are a simplification of the second messengers processes. In Ennis' model transducers and receptor proteins are odor specific. We generalize Ennis' model by introducing transducer mechanisms $T_k$ that can be activated by odor-receptor complexes, and as with odors and receptor proteins we allow receptor proteins and transducers to react in any combination. Receptor proteins and transducer proteins are not required to be odor specific.

The kinetics of the two reactions are given by

$$
\begin{aligned}
A_i + R_j &\rightleftharpoons A_i R_j \\
A_i R_j + T_k &\rightleftharpoons A_i R_j T_k \; .
\end{aligned}
\tag{1}
$$

In a first reaction odor ligands $A_i$ bind to receptor proteins $R_j$ and build odor-receptor complexes $A_i R_j$, which can activate transducer mechanisms $T_k$ in a second reaction.

Affinities $k_{ij}$ and $l_{jk}$ describe the possibility of reactions between odor ligands $A_i$ and receptor proteins $R_j$ or between odor-receptor complexes $A_i R_j$ and transducers $T_k$, respectively. The mass action equations are

$$
\begin{aligned}
[A_i R_j] &= k_{ij}[A_i][R_j] \\
[A_i R_j T_k] &= l_{jk}[A_i R_j][T_k] \; .
\end{aligned}
\tag{2}
$$

The binding of odor-receptor complexes with transducer mechanisms is not dependent on the specific odor which is bound to the receptor protein, i.e. $l_{jk}$ does not depend on $i$. It is only necessary that the receptor protein is bound.

A sensory neuron can now be defined by the total concentration (or amount) of receptor proteins $[\hat{R}]$ and transducers $[\hat{T}]$. The total concentration of either type corresponds to the sum of the free sites and the bound sites:

$$
[\hat{R}_j] = [R_j] + \sum_i [A_i R_j]
\tag{3}
$$

$$
[\hat{T}_k] = [T_k] + \sum_{i,j} [A_i R_j T_k] \; .^{[1]}
\tag{4}
$$

Activated transducer mechanisms may elicit an excitatory or inhibitory effect depending on the kind of ion channel they open. Thus we divide the transducers $T_k$ into two types: inhibitory and excitatory transducers. With

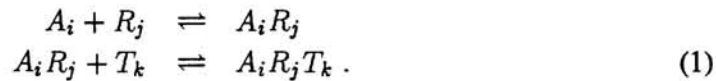

$$
\delta_k = \begin{cases} +1 & \text{, if transducer } T_k \text{ is excitatory} \\ -1 & \text{, if transducer } T_k \text{ is inhibitory} \end{cases}
\tag{5}
$$

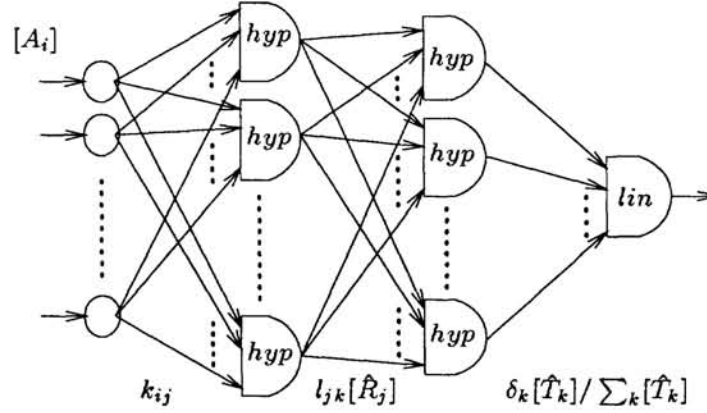

Figure 2: ANN equivalent to the full receptor-transducer model. The input layer corre-
sponds to the concentration of odor ligands $[A_i]$, the first hidden layer to activated receptor
protein types, the second to activated transducer mechanisms. The output neuron computes
the effect E of the sensory cell.

the effect can be set to the sum of all activated excitatory transducers minus the sum of all
inhibitory transducers relative to the total amount of transducers. An additive constant $\theta$ is
used to model spontaneous reactions. With this the effect of an odor can be set to

$$E = \left( \sum_k \delta_k \sum_{i,j} [A_i R_j T_k] \right) \Big/ \left( \sum_k [\hat{T}_k] \right) + \theta \ . \qquad (6)$$

With Eqs.(2,4) and the hyperbolic function $\text{hyp}(x) = x/(1+x)$ the effect $E$ defined in
Eq.(6) can be reformulated to

$$E = \frac{1}{\sum_k [\hat{T}_k]} \sum_k \text{hyp} \left( \sum_{i,j} l_{jk}[A_i R_j] \right) \delta_k [\hat{T}_k] + \theta \ . \qquad (7)$$

Analogously, we eliminate $[A_i R_j]$ and $[R_j]$:

$$E = \frac{1}{\sum_k [\hat{T}_k]} \sum_k \text{hyp} \left( \sum_j l_{jk}[\hat{R}_j] \, \text{hyp} \left( \sum_i k_{ij}[A_i] \right) \right) \delta_k [\hat{T}_k] + \theta \ . \qquad (8)$$

Equation (8) can now be regarded as an ANN with 4 feed-forward layers. The concentrations
of the odor ligands $[A_i]$ represent the input layer, the two hidden layers correspond to
activated receptor proteins and activated transducers, and one output element in layer 4
represents the effect caused by the input odor. The weight between the $i$-th element of the
input layer to the $j$-th element of the first hidden layer is $k_{ij}$ and from there to the $k$-th
neuron of the second hidden layer $l_{jk}[\hat{R}_j]$. The weight from element $k$ of hidden layer 2 to
the output element is $\delta_k [\hat{T}_k]/ \sum_k [\hat{T}_k]$. The adaptive elements of the hidden layers have the
hyperbolic activation functions hyp. The network structure is shown in Figure 2.

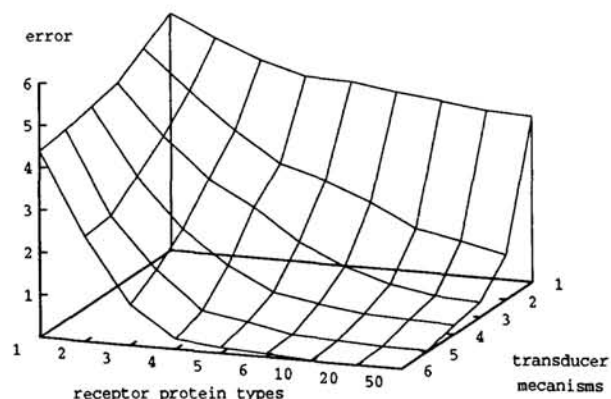

Figure 3: Mean error in spikes per output neuron for the model with different network sizes. Network sizes are varied in the number of receptor protein types and the number of transducing mechanisms.

## 3   SIMULATION RESULTS

Applying learning algorithms like backpropagation or RProp to the model network, it is possible to find parameter settings for optimal (or local optimal) simulations of chemosensory cell responses with given response characteristics. In our simulations the best training results were achieved by using the fast learning algorithm RProp, which is an improved version of backpropagation [Riedmiller & Braun, 1993].

For our simulations we used extracellular recordings made by Akers and Getz from single sensilla placodes of honey bee workers applying different stimuli and their binary mixtures to the antenna (see [Akers & Getz, 1992] for material and methods). The data set for training the ANNs consists of responses of 54 subplacodes to the four odors, geraniol, citral, limonene, linalool, their binary mixtures, and a mixture of all of four odors each at two concentration levels and to a blank stimulus, i.e. 23 responses to different odor stimulations for each subplacode.

In a series of training runs with varying numbers of receptor protein types and transducer types the full model was trained to fit the data set. The networks were able to simulate the responses of the subplacodes, dependent on the network size. The size of the first hidden layer corresponds to the number of receptor protein types ($R$) in the model, the size of the second hidden layer corresponds to the number of transducing mechanisms ($T$).

Figure 3 shows the mean error per output neuron in spikes for all combinations of one to six receptor types and one to six transducer mechanisms and for combinations with ten, twenty and fifty receptor protein types.

The mean response over all subplacode responses is 18.15 spikes. The best results with errors less than two spikes per response were achieved with models with at least three receptor protein types and at least three transducer mechanisms. A model with only two transducer types is not sufficient to simulate the data.

For generalization tests we generated a larger pattern set with our model. This training set

none

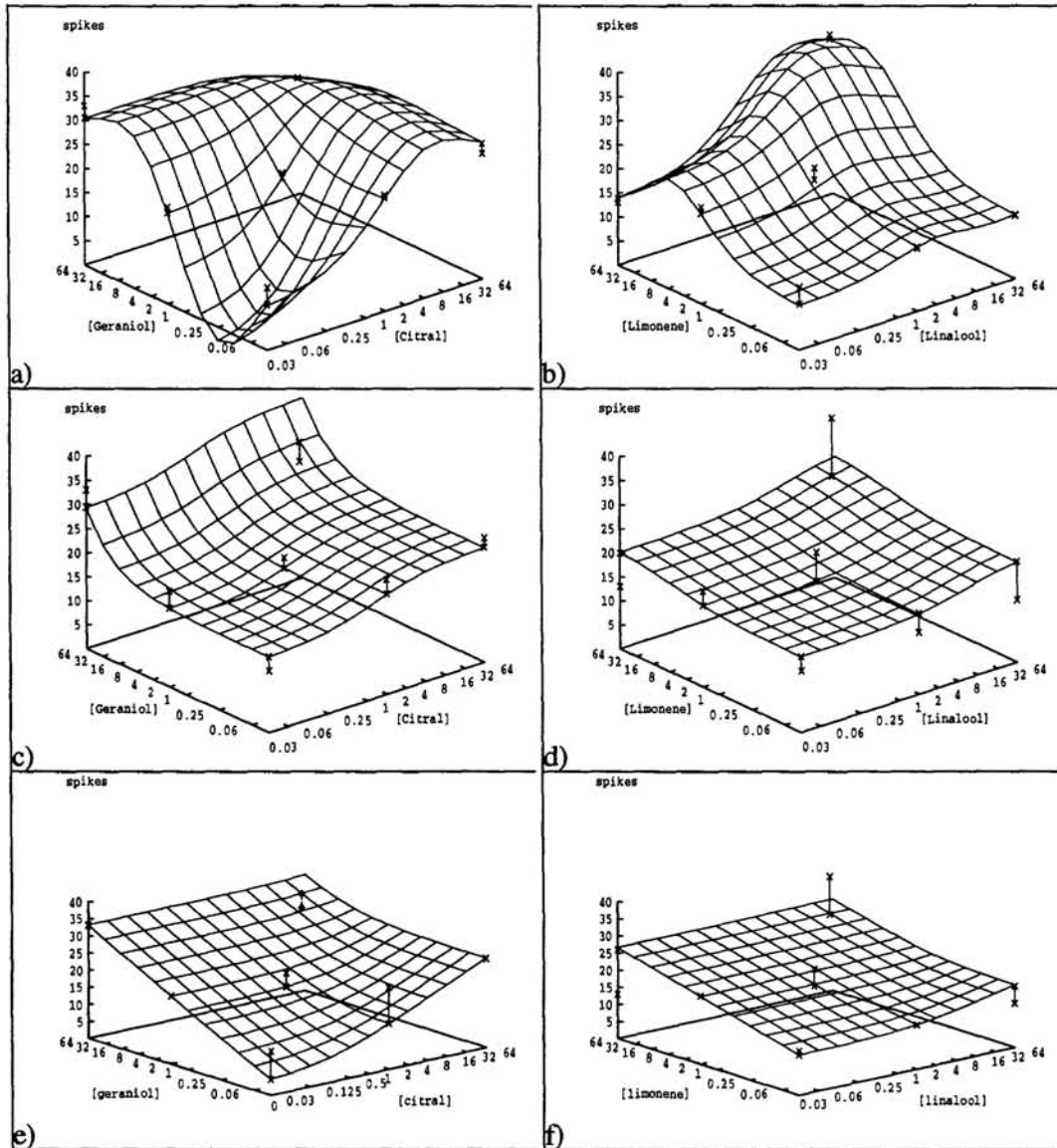

Figure 4: Simulation results of our model (a,b) and the Ennis model (c,d). The responses of simulated sensory cells is given in spikes. The left column (a,c) represents receptor neuron responses to mixtures of geraniol and citral, the right column (b,d) represents sensory cell responses to mixtures of limonene and linalool. The concentrations of the odorants are depicted on a logarithmic scale from $2^{-5}$ to $2^6$ micrograms (0.03 to 64 micrograms). Measurement points and deviations from simulated data are given by crosses in the diagrams.

was divided in a set of 23 training patterns and 88 test patterns. The training set had the same structure as the experimental data. Training of new randomly initialized networks provided a mean error on the test set that was approximately 1.6 times higher than on the training set. An overfitting effect was not observable during the training sequence of 10000

learning epochs.

It is also possible to transform many other models for chemosensory perception into ANNs. We fitted the stimulus summation model and the stimulus substitution model [Carr & Derby, 1986] as well as the models proposed by Ennis [Ennis, 1991]. All of the other models were not able to reproduce the complex response functions observed in honey bee sensory neurons. Some of them are able to simulate synergistic responses to binary mixtures, but none were able to produce inhibitory effects. Figure 4 shows the simulation of a sensory neuron that shows very similar spike rates for the single odors geraniol and citral and to their binary mixture at the same concentration, while the mixture interaction of limonene and linalool shows a strong synergistic effect, i.e. the response to mixture of both odors is much higher than the responses to the single odors. As shown in Figure 4a) and b) our model is able to simulate this behavior, while the Ennis model is not sufficient to show the two different types of interaction for the binary mixtures geraniol-citral and limonene-linalool, as shown in Figure 4c) and d). The error for the Ennis model is greater than four spikes per output neuron and the error for our model with six receptor types and four transducer mechanisms is smaller than one spike per output neuron. The stimulus summation and stimulus substitution model have very similar results as the Ennis model, Figure 4 e) and f) show the simulation of the stimulus summation.

## 4 CONCLUSIONS

Artificial neural networks are a powerful tool for the simulation of the responses of chemosensory cells. The use of ANNs is consistent with theoretical modelings. Many previously proposed models are expressible as ANNs. The new receptor transducer model described in this paper is also expressible as an ANN. The use of learning algorithms is a means to fit parameters for the simulation with given experimental response data. With this method it is possible to create simulation models of chemosensory cells, that can be used in further modelings of olfactory and chemosensory systems.

Applying data from honey bee placode recordings we could also investigate the necessary complexity of chemosensory models. It could be shown that only the full receptor transducer model is able to simulate the complex response characteristics observed in honey bee chemosensory cells. Most other models can show only low synergistic mixture interactions and none of the other models is able to simulate inhibitory effects in odor perception.

The found parameters of the ANN do not have to correspond to physiological entities, such as affinities between molecules. The learning or parameter fitting optimizes the parameters in a way that the difference between experimental data and simulation results is minimized. If there are several solutions to this task, one solution will be found, which might differ from the actual values. But it can be said, that a model is not sufficient if the learning algorithm is not able to fit the experimental data. This implies that the smallest model, which is able to simulate the given data covers the minimum of complexity necessary. For honey bees this means that a competitive receptor transducer model is necessary with at least two transducer mechanisms and three receptor protein types. Any other model, such as the stimulus summation model, the stimulus substitution model and the Ennis model, is not sufficient.

The model is not restricted to insect olfactory receptor neurons and can also be applied to many types of olfactory or gustatory receptor neurons in invertebrates and vertebrates.

## Acknowledgments

We want to thank Pat Akers and Wayne Getz for giving us subplacode response data to train the ANNs used in our model, Heinrich Braun and Wayne Getz for fruitful discussions on our work. This work was supported by grants of the Deutsche Forschungsgemeinschaft (DFG), SPP Physiologie und Theorie neuronaler Netze, and the State of Baden-Württemberg.

## Footnotes

[1] We use the simplification $[\hat{R}_j] = [R_j] + \sum_i [A_i R_j]$ instead of $[\hat{R}_j] = [R_j] + \sum_i [A_i R_j] + \sum_{i,k} [A_i R_j T_k]$, which is sufficient for $[\hat{R}_j] \gg [\hat{T}_k]$, see also [Malaka & Ragg, 1993].

## References

[Akers & Getz, 1992] R.P. Akers & W.M. Getz. A test of identified response classes among olfactory receptor neurons in the honeybee worker. *Chemical Senses*, 17(2):191–209, 1992.

[Akers & Getz, 1993] R.P. Akers & W.M. Getz. Response of olfactory receptor neurons in honey bees to odorants and their binary mixtures. *J. Comp. Physiol. A*, 173:169–185, 1993.

[Breer & Boekhoff, 1992] H. Breer & I. Boekhoff. Second messenger signalling in olfaction. *Current Opinion in Neurobiology*, 2:439–443, 1992.

[Breer *et al.*, 1989] H. Breer, I. Boekhoff, J. Strotmann, K. Raming, & E. Tareilus. Molecular elements of olfactory signal transduction in insect antennae. In D. Schild, editor, *Chemosensory Information Processing*, pages 75–86. Springer, 1989.

[Carr & Derby, 1986] W.E.S. Carr & C.D. Derby. Chemically stimulated feeding behavior in marine animals: the importance of chemical mixtures and the involvement of mixture interactions. *J.Chem.Ecol.*, 12:987–1009, 1986.

[Ennis, 1991] D.M. Ennis. Molecular mixture models based on competitive and non-competitive agonism. *Chemical Senses*, 16(1):1–17, 1991.

[Malaka & Ragg, 1993] R. Malaka & T. Ragg. Models for chemosensory receptors: An approach using artificial neural networks. Interner Bericht 18/93, Institut für Logik, Komplexität und Deduktionssysteme, Universität Karlsruhe, 1993.

[Riedmiller & Braun, 1993] M. Riedmiller & H. Braun. A direct adaptive method for faster backpropagation learning: The rprop algorithm. In *Proceedings of the ICNN*, 1993.

[Shepherd, 1991] G.M. Shepherd. Computational structure of the olfactory system. In J.L. Davis & H. Eichenbaum, editors, *Olfaction — A Model System for Computational Neuroscience*, chapter 1, pages 3–41. MIT Press, 1991.

[Vogt *et al.*, 1989] R.G. Vogt, R. Rybczynski, & M.R. Lerner. The biochemistry of odorant reception and transduction. In D. Schild, editor, *Chemosensory Information Processing*, pages 33–76. Springer, 1989.